# A silicon primitive for competitive learning

**David Hsu**             **Miguel Figueroa**             **Chris Diorio**

Computer Science and Engineering
The University of Washington
114 Sieg Hall, Box 352350
Seattle, WA 98195-2350 USA
hsud, miguel, diorio@cs.washington.edu

## Abstract

Competitive learning is a technique for training classification and clustering networks. We have designed and fabricated an 11-transistor primitive, that we term an automaximizing bump circuit, that implements competitive learning dynamics. The circuit performs a similarity computation, affords nonvolatile storage, and implements simultaneous local adaptation and computation. We show that our primitive is suitable for implementing competitive learning in VLSI, and demonstrate its effectiveness in a standard clustering task.

## 1  Introduction

Competitive learning is a family of neural learning algorithms that has proved useful for training many classification and clustering networks [1]. In these networks, a neuron's synaptic weight vector typically represents a tight cluster of data points. Upon presentation of a new input to the network, the neuron representing the closest cluster adapts its weight vector, decreasing the difference between the weight vector and present input. Details on this adaptation vary for different competitive learning rules, but the general functionality of the synapse is preserved across various competitive learning networks. These functions are weight storage, similarity computation, and competitive learning dynamics.

Many VLSI implementations of competitive learning have been reported in the literature [2]. These circuits typically use digital registers or capacitors for weight storage. Digital storage is expensive in terms of die area and power consumption; capacitive storage typically requires a refresh scheme to prevent weight decay. In addition, these implementations require separate computation and weight-update phases, increasing complexity. More importantly, neural networks built with these circuits typically do not adapt during normal operation.

Synapse transistors [3][4] address the problems raised in the previous paragraph. These devices use the floating-gate technology to provide nonvolatile analog storage and local adaptation in silicon. The adaptation mechanisms do not perturb the operation of the device, thus enabling simultaneous adaptation and computation. Unfortunately, the adaptation mechanisms provide dynamics that are difficult to translate

into existing neural-network learning rules. Allen et. al. [5] proposed a silicon competitive learning synapse that used floating gate technology in the early 90's. However, that approach suffers from asymmetric adaptation due to separate mechanisms for increasing and decreasing weight values. In addition, they neither characterized the adaptation dynamics of their device, nor demonstrated competitive learning with their device.

We present a new silicon primitive, the *automaximizing bump circuit*, that uses synapse transistors to implement competitive learning in silicon. This 11-transistor circuit computes a similarity measure, provides nonvolatile storage, implements local adaptation, and performs simultaneous adaptation and computation. In addition, the circuit naturally exhibits competitive learning dynamics. In this paper, we derive the properties of the automaximizing bump circuit directly from the physics of synapse transistors, and corroborate our analysis with data measured from a chip fabricated in a 0.35μm CMOS process. In addition, experiments on a competitive learning circuit, and software simulations of the learning rule, show that this device provides a suitable primitive for competitive learning.

## 2 Synapse transistors

The automaxmizing bump circuit's behavior depends on the storage and adaptation properties of synapse transistors. Therefore this section briefly reviews these devices. A synapse transistor comprises a floating-gate MOSFET, with a control gate capacitively coupled to the floating gate, and an associated tunneling implant. The transistor uses floating-gate charge to implement a nonvolatile analog memory, and outputs a source current that varies with both the stored value and the control-gate voltage. The synapse uses two adaptation mechanisms: Fowler-Nordheim tunneling [6] increases the stored charge; impact-ionized hot-electron injection (IHEI) [7] decreases the charge. Because tunneling and IHEI can both be active during normal transistor operation, the synapse enables simultaneous adaptation and computation.

A voltage difference between the floating gate and the tunneling implant causes electrons to tunnel from the floating gate, through gate oxide, to the tunneling implant. We can approximate this current (with respect to fixed tunneling and floating-gate voltages, $V_{tun0}$ and $V_{g0}$) as [4]:

$$I_{tun} = I_{tun0}e^{(\Delta V_{tun} - \Delta V_{fg})/V_\chi} \tag{1}$$

where $I_{tun0}$ and $V_\chi$ are constants that depend on $V_{tun0}$ and $V_{g0}$, and $\Delta V_{tun}$ and $\Delta V_g$ are deviations of the tunneling and floating gate voltages from these fixed levels.

IHEI adds electrons to the floating gate, decreasing its stored charge. The IHEI current increases with the transistor's source current and drain-to-source voltage; over a small drain-voltage range, we model this dependence as [3][4]:

$$I_{inj} = I_{inj0}I_s^{1-U_t/V_\gamma}e^{V_{sd}/V_\gamma} \tag{2}$$

where the constant $V_\gamma$ depends on the VLSI process, and $U_t$ is the thermal voltage.

## 3 Automaximizing bump circuit

The automaximizing bump circuit (Fig. 1) is an adaptive version of the classic bump-antibump circuit [8]. It uses synapse transistors to implement the three essential functions of a competitive learning synapse: storage of a weight value $\mu$, computation of a similarity measure between the input and $\mu$, and the ability to move $\mu$ closer to the input. Both circuits take two inputs, $V_1$ and $V_2$, and generate three cur-

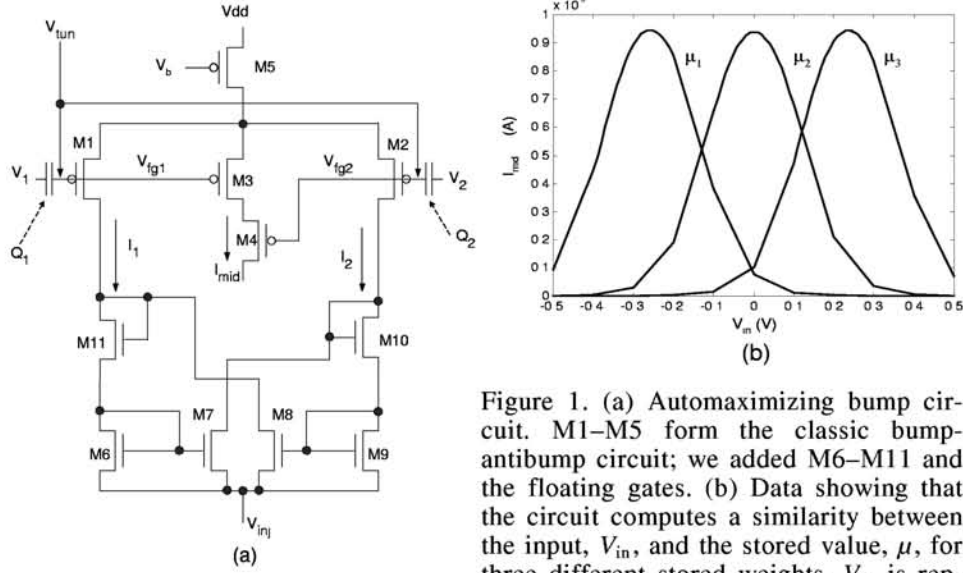

**(a)**

**(b)**

Figure 1. (a) Automaximizing bump circuit. M1–M5 form the classic bump-antibump circuit; we added M6–M11 and the floating gates. (b) Data showing that the circuit computes a similarity between the input, $V_{in}$, and the stored value, $\mu$, for three different stored weights. $V_{in}$ is represented as $V_1 = +V_{in}/2$, $V_2 = -V_{in}/2$.

rents. The two outside currents, $I_1$ and $I_2$, are a measure of the dissimilarity between the two inputs; the center current, $I_{mid}$, is a measure of their similarity:

$$I_{mid} = I_b(1 + \lambda \cosh^2(\kappa \Delta V))^{-1} \qquad (3)$$

where $\lambda$ and $\kappa$ are process and design-dependent parameters, $\Delta V$ is the voltage difference between $V_1$ and $V_2$, and $I_b$ is a bias current. $I_{mid}$ is symmetric with respect to the difference between $V_1$ and $V_2$, and approximates a Gaussian centered at $\Delta V = 0$.

We augment the bump-antibump circuit by adding floating gates and tunneling junctions to M1-M5, turning them into synapse transistors; M1 and M3 share the same floating gate and tunneling junction, as do M2 and M4. We also add transistors M6-M11 to control IHEI. For convenience, we will refer to our new circuit merely as a *bump circuit*. The charge stored on the bump circuit's floating gates, $Q_1$ and $Q_2$, shift $I_{mid}$'s peak away from $\Delta V = 0$ by an amount determined by their difference. We interpret this difference as the weight, $\mu$, stored by the circuit, and interpret $I_{mid}$ as a similarity measure between the circuit's input and stored weight.

Tunneling and IHEI adapt the bump circuit's weight. The circuit is *automaximizing* because tunneling and IHEI naturally tune the peak of $I_{mid}$ to coincide with the present input. This high-level behavior coincides with the dynamics of competitive learning; both act to decrease the difference between a stored weight and the applied input. Therefore, no explicit computation of the direction or magnitude of weight updates is necessary—the circuit naturally performs these computations for us. Consequently, we only need to indicate when the circuit should adapt, not how it does adapt. Applying ~10V to $V_{tun}$ and ~0V to $V_{inj}$ activates adaptation. Applying <8V to $V_{tun}$ and >2V to $V_{inj}$ deactivates adaptation.

## 3.1 Weight storage

The bump circuit's weight value derives directly from the charge on its floating-gates. A synapse transistor's floating-gate charge looks, for all practical purposes,

like a voltage source, $V_s$, applied to the control gate. This voltage source has a value $V_s = Q/C_{in}$, where $C_{in}$ is the control-gate to floating-gate coupling capacitance and $Q$ is the floating gate charge. We encode the input to the bump circuit, $V_{in}$, as a differential signal: $V_1 = V_{in}/2$; and $V_2 = -V_{in}/2$ (similar results will follow for any symmetric encoding of $V_{in}$). As a result, $I_{mid}$ computes the similarity between the two floating-gate voltages: $V_{fg1} = V_{s1} + V_{in}/2$, and $V_{fg2} = V_{s2} - V_{in}/2$ where $V_{s1}$ and $V_{s2}$ are the voltages due to the charge stored on the floating gates. We define the bump circuit's weight, $\mu$, as:

$$\mu = V_{s2} - V_{s1} \qquad (4)$$

This weight corresponds to the value of $V_{in}$ that equalizes the two floating-gate voltages (and maximizes $I_{mid}$). Part (b) of Fig. 1 shows the bump circuit's $I_{mid}$ output for three weight values, as a function of the differential input. We see that different stored values change the location of the peak, but do not change the shape of the bump. Because floating gate charge is nonvolatile, the weight is also nonvolatile.

The differential encoding of the input makes the bump circuit's adaptation symmetric with respect to $(V_{in} - \mu)$. Without loss of generality, we can represent $V_{in}$ as:

$$V_{in} = V_{s2} - V_{s1} + (V_{in} - \mu) \qquad (5)$$

If we apply $V_{in}/2$ and $-V_{in}/2$ to the two input terminals, we arrive at the following two floating-gate voltages:

$$V_{fg1} = (V_{s2} + V_{s1} + V_{in} - \mu)/2 \qquad (6)$$

$$V_{fg2} = (V_{s2} + V_{s1} - V_{in} + \mu)/2 \qquad (7)$$

By reversing the sign of $(V_{in} - \mu)$, we obtain the same floating-gate voltages on the opposite terminals. Because the floating gate voltages are independent of the sign of $(V_{in} - \mu)$, the bump circuit's learning rule is symmetric with respect to $(V_{in} - \mu)$.

### 3.2 Adaptation

We now explore the bump circuit's adaptation dynamics. We define $\Delta V_{fg} = V_{fg1} - V_{fg2}$. From Eqs. 4–7, we can see that $V_{in} - \mu = \Delta V_{fg}$. Consequently, the learning rate, $d\mu/dt$, is equivalent to $-d\Delta V_{fg}/dt$. In our subsequent derivations, we consider only positive $\Delta V_{fg}$, because adaptation is symmetric (albeit with a change of sign). We show complete derivations of the equations in this section in [9].

Tunneling causes adaptation by decreasing the difference between the floating-gate voltages $V_{fg1}$ and $V_{fg2}$. Electron tunneling increases the voltage of both floating gates, but, because tunneling increases exponentially with smaller floating-gate voltages (see Eq. 1), tunneling decreases the difference. Assuming that M1's floating gate voltage is lower than M2's, the change in $\Delta V_{fg}$ due to electron tunneling is:

$$d\Delta V_{fg}/dt = -(I_{tun1} - I_{tun2})/C_{fg} \qquad (8)$$

We substitute Eq. 1 into Eq. 8 and solve for the tunneling learning rule:

$$d\Delta V_{fg}/dt = -I_{t0}e^{(\Delta V_{tun} - \Delta V_0)/V_\chi} \sinh((\Delta V_{fg} - \phi)/2V_\chi) \qquad (9)$$

where $I_{t0} = I_{tun0}/C_{fg}$, $V_\chi$ is a model constant, $\Delta V_0 = (\Delta V_{fg1} + \Delta V_{fg2})/2$, and $\phi$ models the tunneling mismatch between synapse transistors. This rule depends on three factors:

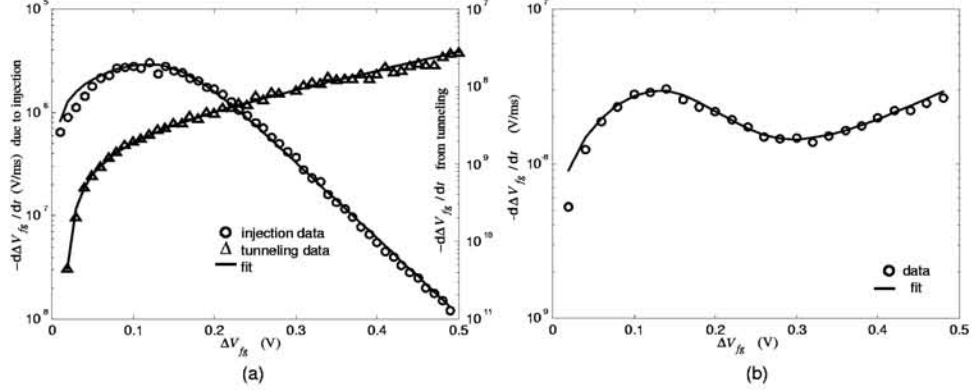

Figure 2. (a) Measured adaptation rates, due to tunneling and IHEI, along with fits from Eqs.9 and 11. (b) Composite adaptation rate, along with a fit from (12). We slowed the IHEI adaptation rate (by using a higher $V_{inj}$), compared with the data from part (a), to cause better matching between tunneling and IHEI.

a controllable learning rate, $\Delta V_{tun}$; the difference between $V_{in}$ and $\mu$, $\Delta V_{fg}$; and the average floating gate voltage, $\Delta V_0$.

The circuit also uses IHEI to decrease $\Delta V_{fg}$. We bias the bump circuit so that only transistors M1 and M2 exhibit IHEI. According to Eq.2, IHEI depends linearly on a transistor's source current, but exponentially on its source-to-drain voltage. Consequently, we decrease $\Delta V_{fg}$ by controlling the drain voltages at M1 and M2. Coupled current mirrors (M6–M7 and M8–M9) at the drains of M1 and M2, simultaneously raise the drain voltage of the transistor that is sourcing a larger current, and lower the drain voltage of the transistor that is sourcing a smaller current. The transistor with the smaller source current will experience a larger $V_{sd}$, and thus exponentially more IHEI, causing its source current to rapidly increase. Diodes (M10 and M11) further increase the drain voltage of the transistor with the larger current, further reducing its IHEI. The net effect is that IHEI acts to equalize the currents, and, likewise, the floating gate voltages. Recently Hasler proposed a similar method for controlling IHEI in a floating gate differential pair [4].

Assuming $I_1 > I_2$, the change in $\Delta V_{fg}$ due to IHEI is:

$$d\Delta V_{fg} / dt = -(I_{inj2} - I_{inj1}) / C_{fg} \qquad (10)$$

We expand the learning rule by substituting Eq.2 into Eq.10. To compute values for the drain voltages of $M_1$ and $M_2$, we assume that all of $I_1$ flows through M11 and all of $I_2$ flows through M7. The IHEI learning rule is given below:

$$d\Delta V_{fg} / dt = -I_{j0}e^{\zeta\Delta V_0}(e^{\tau V_{inj}}\Phi_1(\Delta V_{fg}) - e^{\eta V_{inj}}\Phi_2(\Delta V_{fg})) \qquad (11)$$

where $I_{j0} = I_{inj0}/C_{fg}$, $\tau = -2/\kappa V_\gamma$, $\eta = -1/V_\gamma$, and $\zeta = \kappa/V_\gamma$. $\Phi_1$ and $\Phi_2$ are given by:

$$\Phi_1(\Delta V_{fg}) = ((I_b - I_{mid})/2\cosh(\kappa\Delta V_{fg}/2U_t))^{1-2U_t/\kappa V_\gamma}e^{-\omega\Delta V_{fg}} \qquad (12)$$

$$\Phi_2(\Delta V_{fg}) = ((I_b - I_{mid})/2\cosh(\kappa\Delta V_{fg}/2U_t))e^{-\sigma\Delta V_{fg}}(1 - e^{-\kappa\Delta V_{fg}/U_t})^{-U_t/V_\gamma} \qquad (13)$$

where $\sigma = (1 - U_t/V_\gamma)\kappa/2U_t$, and $\omega = \kappa/2U_t - \kappa/2V_\gamma - 1/V_\gamma$. Like tunneling, the IHEI rule depends on three factors: a controllable learning rate, $V_{inj}$; the difference between $V_{in}$ and $\mu$, $\Delta V_{fg}$; and $\Delta V_0$. Part (a) of Fig. 2 shows measurements of $d\Delta V_{fg}/dt$ versus $\Delta V_{fg}$ due to tunneling and IHEI, along with fits to Eqs.9 and 11 respectively.

IHEI and tunneling facilitate adaptation by adding and removing charge from the floating gates, respectively. Isolated, any of these mechanisms will eventually drive the bump circuit out of its operating range. In order to obtain useful adaptation, we need to activate both mechanisms at the same time. There is an added benefit to combining tunneling and IHEI: Part (a) Fig 2 shows that tunneling acts more strongly for smaller values of $\Delta V_{\text{fg}}$, while IHEI shows the opposite behavior. The mechanisms complement each other, providing adaptation over more than a 1 V range in $\Delta V_{\text{fg}}$. We combine Eq. 9 and Eq. 11 to derive the bump learning rule:

$$-d\Delta V_{\text{fg}}/dt = I_{t0}e^{(\Delta V_{\text{tun}}-\Delta V_0)/V_\chi}\sinh((\Delta V_{\text{fg}}-\phi)/2V_\chi)+I_{j0}e^{\zeta\Delta V_0}(e^{\tau V_{\text{inj}}}\Phi_1(\Delta V_{\text{fg}})-e^{\eta V_{\text{inj}}}\Phi_2(\Delta V_{\text{fg}})) \quad (14)$$

Part (b) of Fig. 2 illustrates the composite weight-update dynamics. When $\Delta V_{\text{fg}}$ is small, adaptation is primarily driven by IHEI, while tunneling dominates for larger values of $\Delta V_{\text{fg}}$.

The bump learning rule is unlike any learning rule that we have found in the literature. Nevertheless, it exhibits several desirable properties. First, it naturally moves the bump circuit's weight towards the present input. Second, the weight update is symmetric with respect to the difference between the stored value and the present input. Third, we can vary the weight-update rate over many orders of magnitude by adjusting $V_{\text{tun}}$ and $V_{\text{inj}}$. Finally, because the bump circuit uses synapse transistors to perform adaptation, the circuit can adapt during normal operation.

## 4 Competitive learning with bump circuits

We summarize the results of simulations of the bump learning rule and also results from a competitive learning circuit fabricated in the TSMC 0.35 μm process below. For further details consult [9]. We first compared the performance of a software neural network on a standard clustering task, using the bump learning rule (fitted to data from Fig. 2), and a basic competitive learning rule (learning rate $\rho=0.01$):

$$d\vec{\mu}/dt = \rho \times (\vec{V}_{in} - \vec{\mu}) \quad (15)$$

We trained both networks on data drawn from a mixture of 32 Gaussians, in a 32-dimensional space. The Gaussian means were drawn from the interval [0,1] and the covariance matrix was the diagonal matrix 0.1*$\mathbf{I}$. On an input presentation, the network updated the weight vector of the closest neuron using either the bump learning rule, or Eq. 15. We measured the performance of the two learning rules by evaluating the coding error of each trained network, on a test set drawn from the same distribution as the training data. The coding error is the sum of the squared distances between each test point and its closest neuron. Part (a) of Fig. 3 shows that the bump circuit's rule performs favorably with the hard competitive learning rule.

Our VLSI circuit (Part (b) of Fig. 3) comprised two neurons with a one-dimensional input (a neuron was a single bump circuit), and a feedback network to control adaptation. The feedback network comprised a winner-take-all (WTA) [10] that detected which bump was closest to the present input, and additional circuitry [9] that generated $V_{\text{tun}}$ and $V_{\text{inj}}$ from the WTA output. We tested this circuit on a clustering task, to learn the centers of a mixture of two Gaussians. In part (c) of Fig. 3, we compare the performance of our circuit with a simulated neural network using Eq. 15. The VLSI circuit performed comparably with the neural network, demonstrating that our bump circuit, in conjunction with simple feedback mechanisms, can implement competitive learning in VLSI. We can generalize the circuitry to multiple dimensions (multiple bump circuits per neuron) and multiple neurons; each neuron only requires one $V_{\text{tun}}$ and $V_{\text{inj}}$ signal.

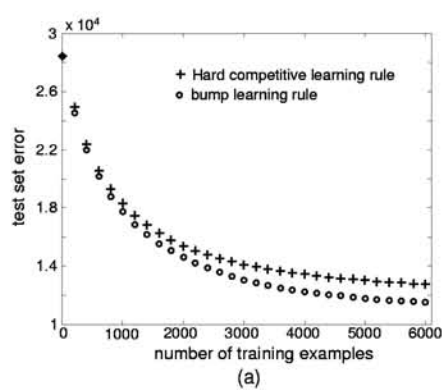

(a)

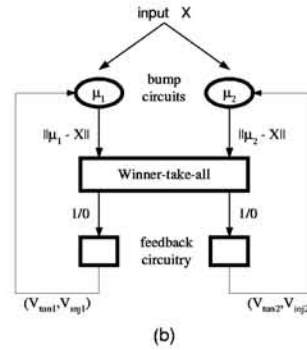

(b)

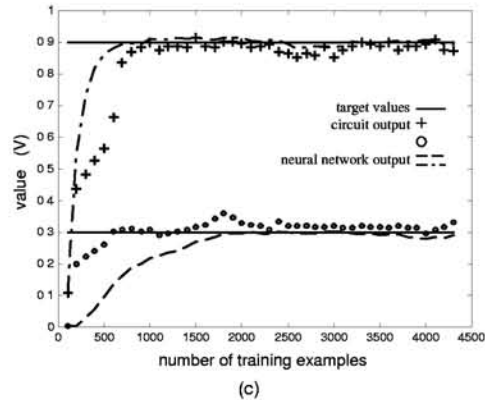

(c)

Figure 3. (a) Comparison of a neural network using the bump learning rule versus a standard competitive learning rule. We drew the training data from a mixture of thirty-two Gaussians, and averaged the results over ten trials. (b) A competitive learning circuit. (c) Performance of a competitive learning circuit versus a neural network for learning a mixture of two Gaussians.

## Acknowledgements

This work was supported by the NSF under grants BES 9720353 and ECS 9733425, and by a Packard Foundation Fellowship.

## References

[1] M.A. Arbib (ed.), *The Handbook of Brain Theory and Neural Networks*, Cambridge, MA: The MIT Press, 1995.

[2] H.C. Card, D.K. McNeill, and C.R. Schneider, "Analog VLSI circuits for competitive learning networks", in *Analog Integrated Circuits and Signal Processing,* 15, pp. 291-314, 1998.

[3] C. Diorio, "A *p*-channel MOS synapse transistor with self-convergent memory writes", *IEEE Transactions on Electron Devices*, vol. 47, no. 2, pp 464-472, 2000.

[4] P. Hasler, ``Continuous-Time Feedback in Floating-Gate MOS Circuits," to appear in *IEEE Transactions on Circuits and Systems II*, Feb. 2001

[5] T. Allen et. al, "Electrically adaptable neural network with post-processing circuitry," U.S. Patent No. 5,331,215, issued July 19, 1994.

[6] M. Lenzlinger and E.H. Snow, "Fowler–Nordheim tunneling into thermally grown $SiO_2$", *Journal of Applied Physics*, vol. 40(1), pp. 278-283, 1969.

[7] E. Takeda, C. Yang, and A. Miura-Hamada, *Hot Carrier Effects in MOS Devices*, San Diego, CA: Academic Press, 1995.

[8] T. Delbruck, "Bump circuits for computing similarity and dissimilarity of analog voltages", CNS Memo 26, California Institute of Technology, 1993.

[9] D. Hsu, M. Figueroa, and C. Diorio, "A silicon primitive for competitive learning," UW CSE Technical Report no. 2000-07-01, 2000.

[10] J. Lazzaro, S. Ryckebusch, M.A. Mahowald, and C.A. Mead, "Winner-take-all networks of O(n) complexity", in *Advances in Neural Information Processing Systems,* San Mateo, CA: Morgan Kaufman, vol. 1, pp 703-711, 1989.
